# Correcting sample selection bias in maximum entropy density estimation

**Miroslav Dudík, Robert E. Schapire**
Princeton University
Department of Computer Science
35 Olden St, Princeton, NJ 08544
{mdudik,schapire}@princeton.edu

**Steven J. Phillips**
AT&T Labs − Research
180 Park Ave, Florham Park, NJ 07932
phillips@research.att.com

## Abstract

We study the problem of maximum entropy density estimation in the presence of known sample selection bias. We propose three bias correction approaches. The first one takes advantage of unbiased sufficient statistics which can be obtained from biased samples. The second one estimates the biased distribution and then factors the bias out. The third one approximates the second by only using samples from the sampling distribution. We provide guarantees for the first two approaches and evaluate the performance of all three approaches in synthetic experiments and on real data from species habitat modeling, where maxent has been successfully applied and where sample selection bias is a significant problem.

## 1  Introduction

We study the problem of estimating a probability distribution, particularly in the context of species habitat modeling. It is very common in distribution modeling to assume access to independent samples from the distribution being estimated. In practice, this assumption is violated for various reasons. For example, habitat modeling is typically based on known occurrence locations derived from collections in natural history museums and herbariums as well as biological surveys [1, 2, 3]. Here, the goal is to predict the species' distribution as a function of climatic and other environmental variables. To achieve this in a statistically sound manner using current methods, it is necessary to assume that the sampling distribution and species distributions are not correlated. In fact, however, most sampling is done in locations that are easier to access, such as areas close to towns, roads, airports or waterways [4]. Furthermore, the independence assumption may not hold since roads and waterways are often correlated with topography and vegetation which influence species distributions. New unbiased sampling may be expensive, so much can be gained by using the extensive existing biased data, especially since it is becoming freely available online [5].

Although the available data may have been collected in a biased manner, we usually have some information available about the nature of the bias. For instance, in the case of habitat modeling, some factors influencing the sampling distribution are well known, such as distance from roads, towns, etc. In addition, a list of visited sites may be available and viewed as a sample of the sampling distribution itself. If such a list is not available, the set of sites where any species from a large group has been observed may be a reasonable approximation of all visited locations.

In this paper, we study probability density estimation under sample selection bias. We

assume that the sampling distribution (or an approximation) is known during *training*, but we require that unbiased models not use any knowledge of sample selection bias during *testing*. This requirement is vital for habitat modeling where models are often applied to a different region or under different climatic conditions. To our knowledge this is the first work addressing sample selection bias in a statistically sound manner and in a setup suitable for species habitat modeling from presence-only data.

We propose three approaches that incorporate sample selection bias in a common density estimation technique based on the principle of maximum entropy (maxent). Maxent with $\ell_1$-regularization has been successfully used to model geographic distributions of species under the assumption that samples are unbiased [3]. We review $\ell_1$-regularized maxent with unbiased data in Section 2, and give details of the new approaches in Section 3.

Our three approaches make simple modifications to unbiased maxent and achieve analogous provable performance guarantees. The first approach uses a bias correction technique similar to that of Zadrozny et al. [6, 7] to obtain unbiased confidence intervals from biased samples as required by our version of maxent. We prove that, as in the unbiased case, this produces models whose log loss approaches that of the best possible Gibbs distribution (with increasing sample size).

In contrast, the second approach we propose first estimates the biased distribution and then factors the bias out. When the target distribution is a Gibbs distribution, the solution again approaches the log loss of the target distribution. When the target distribution is not Gibbs, we demonstrate that the second approach need not produce the optimal Gibbs distribution (with respect to log loss) even in the limit of infinitely many samples. However, we prove that it produces models that are almost as good as the best Gibbs distribution according to a certain Bregman divergence that depends on the selection bias. In addition, we observe good empirical performance for moderate sample sizes. The third approach is an approximation of the second approach which uses samples from the sampling distribution instead of the distribution itself.

One of the challenges in studying methods for correcting sample selection bias is that unbiased data sets, though not required during training, are needed as test sets to evaluate performance. Unbiased data sets are difficult to obtain — this is the very reason why we study this problem! Thus, it is almost inevitable that synthetic data must be used. In Section 4, we describe experiments evaluating performance of the three methods. We use both fully synthetic data, as well as a biological dataset consisting of a biased training set and an independently collected reasonably unbiased test set.

**Related work.** Sample selection bias also arises in econometrics where it stems from factors such as attrition, nonresponse and self selection [8, 9, 10]. It has been extensively studied in the context of linear regression after Heckman's seminal paper [8] in which the bias is first estimated and then a transform of the estimate is used as an additional regressor.

In the machine learning community, sample selection bias has been recently considered for classification problems by Zadrozny [6]. Here the goal is to learn a decision rule from a biased sample. The problem is closely related to cost-sensitive learning [11, 7] and the same techniques such as resampling or differential weighting of samples apply.

However, the methods of the previous two approaches do not apply directly to density estimation where the setup is "unconditional", i.e. there is no dependent variable, or, in the classification terminology, we only have access to positive examples, and the cost function (log loss) is unbounded. In addition, in the case of modeling species habitats, we face the challenge of sample sizes that are very small (2–100) by machine learning standards.

## 2   Maxent setup

In this section, we describe the setup for unbiased maximum entropy density estimation and review performance guarantees. We use a relaxed formulation which will yield an $\ell_1$-regularization term in our objective function.

The goal is to estimate an unknown *target distribution* $\pi$ over a known *sample space* $\mathcal{X}$ based on *samples* $x_1, \ldots, x_m \in \mathcal{X}$. We assume that samples are independently distributed according to $\pi$ and denote the *empirical distribution* by $\tilde{\pi}(x) = |\{1 \leq i \leq m : x_i = x\}|/m$. The structure of the problem is specified by real valued functions $f_j : \mathcal{X} \to \mathbb{R}$, $j = 1, \ldots, n$, called *features* and by a distribution $q_0$ representing a *default estimate*. We assume that features capture all the relevant information available for the problem at hand and $q_0$ is the distribution we would choose if we were given no samples. The distribution $q_0$ is most often assumed uniform.

For a limited number of samples, we expect that $\tilde{\pi}$ will be a poor estimate of $\pi$ under any reasonable distance measure. However, empirical averages of features will not be too different from their expectations with respect to $\pi$. Let $p[f]$ denote the expectation of a function $f(x)$ when $x$ is chosen randomly according to distribution $p$. We would like to find a distribution $p$ which satisfies

$$|p[f_j] - \tilde{\pi}[f_j]| \leq \beta_j \ \text{ for all } 1 \leq j \leq n, \tag{1}$$

for some estimates $\beta_j$ of deviations of empirical averages from their expectations. Usually there will be infinitely many distributions satisfying these constraints. For the case when the default distribution $q_0$ is uniform, the maximum entropy principle tells us to choose the distribution of maximum entropy satisfying these constraints. In general, we should minimize the relative entropy from $q_0$. This corresponds to choosing the distribution that satisfies the constraints (1) but imposes as little additional information as possible when compared with $q_0$. Allowing for asymmetric constraints, we obtain the formulation

$$\min_{p \in \Delta} \text{RE}(p \parallel q_0) \text{ subject to } \forall 1 \leq j \leq n : \ a_j \leq p[f_j] \leq b_j. \tag{2}$$

Here, $\Delta \subseteq \mathbb{R}^{\mathcal{X}}$ is the simplex of probability distributions and $\text{RE}(p \parallel q)$ is the relative entropy (or Kullback-Leibler divergence) from $q$ to $p$, an information theoretic measure of difference between the two distributions. It is non-negative, equal to zero only when the two distributions are identical, and convex in its arguments.

Problem (2) is a convex program. Using Lagrange multipliers, we obtain that the solution takes the form

$$q_{\boldsymbol{\lambda}}(x) = q_0(x)e^{\boldsymbol{\lambda} \cdot \boldsymbol{f}(x)}/Z_{\boldsymbol{\lambda}} \tag{3}$$

where $Z_{\boldsymbol{\lambda}} = \sum_x q_0(x)e^{\boldsymbol{\lambda} \cdot \boldsymbol{f}(x)}$ is the normalization constant. Distributions $q_{\boldsymbol{\lambda}}$ of the form (3) will be referred to as $q_0$-Gibbs or just Gibbs when no ambiguity arises.

Instead of solving (2) directly, we solve its dual:

$$\min_{\boldsymbol{\lambda} \in \mathbb{R}^n} \left( \log Z_{\boldsymbol{\lambda}} - \tfrac{1}{2}\textstyle\sum_j (b_j + a_j)\lambda_j + \tfrac{1}{2}\textstyle\sum_j (b_j - a_j)|\lambda_j| \right). \tag{4}$$

We can choose from a range of general convex optimization techniques or use some of the algorithms in [12]. For the symmetric case when

$$[a_j, b_j] = \left[ \tilde{\pi}[f_j] - \beta_j, \ \tilde{\pi}[f_j] + \beta_j \right], \tag{5}$$

the dual becomes

$$\min_{\boldsymbol{\lambda} \in \mathbb{R}^n} \left( -\tilde{\pi}[\log q_{\boldsymbol{\lambda}}] + \textstyle\sum_j \beta_j|\lambda_j| \right). \tag{6}$$

The first term is the *empirical log loss* (negative log likelihood), the second term is an $\ell_1$-*regularization*. Small values of log loss mean a good fit to the data. This is balanced by regularization forcing simpler models and hence preventing overfitting.

When all the primal constraints are satisfied by the target distribution $\pi$ then the solution $\hat{q}$ of the dual is guaranteed to be not much worse an approximation of $\pi$ than the best Gibbs distribution $q^*$. More precisely:

**Theorem 1** (Performance guarantees, Theorem 1 of [12])**.** *Assume that the distribution $\pi$ satisfies the primal constraints* (2)*. Let $\hat{q}$ be the solution of the dual* (4)*. Then for an arbitrary Gibbs distribution $q^* = q_{\boldsymbol{\lambda}*}$*

$$\text{RE}(\pi \parallel \hat{q}) \leq \text{RE}(\pi \parallel q^*) + \textstyle\sum_j (b_j - a_j)|\lambda_j^*|.$$

Algorithm 1: DEBIASAVERAGES.

Table 1: *Example 1.* Comparison of distributions $q^*$ and $q^{**}$ minimizing $\mathrm{RE}(\pi \parallel q_\lambda)$ and $\mathrm{RE}(\pi s \parallel q_\lambda s)$.

| $x$ | $\boldsymbol{f}(x)$ | $\pi(x)$ | $s(x)$ | $\pi s(x)$ | $q^*(x)$ | $q^{**}s(x)$ | $q^{**}(x)$ |
|---|---|---|---|---|---|---|---|
| 1 | (0, 0) | 0.4 | 0.4 | 0.64 | 0.25 | 0.544 | 0.34 |
| 2 | (0, 1) | 0.1 | 0.4 | 0.16 | 0.25 | 0.256 | 0.16 |
| 3 | (1, 0) | 0.1 | 0.1 | 0.04 | 0.25 | 0.136 | 0.34 |
| 4 | (1, 1) | 0.4 | 0.1 | 0.16 | 0.25 | 0.064 | 0.16 |

When features are bounded between 0 and 1, the symmetric box constraints (5) with $\beta_j = O(\sqrt{(\log n)/m})$ are satisfied with high probability by Hoeffding's inequality and the union bound. Then the relative entropy from $\hat{q}$ to $\pi$ will not be worse than the relative entropy from any Gibbs distribution $q^*$ to $\pi$ by more than $O(\|\boldsymbol{\lambda^*}\|_1 \sqrt{(\log n)/m})$.

In practice, we set

$$\beta_j = \left(\beta/\sqrt{m}\right) \cdot \min\left\{\tilde{\sigma}[f_j], \sigma_{\max}[f_j]\right\} \tag{7}$$

where $\beta$ is a tuned constant, $\tilde{\sigma}[f_j]$ is the sample deviation of $f_j$, and $\sigma_{\max}[f_j]$ is an upper bound on the standard deviation, such as $(\max_x f_j(x) - \min_x f_j(x))/2$. We refer to this algorithm for unbiased data as UNBIASEDMAXENT.

## 3   Maxent with sample selection bias

In the biased case, the goal is to estimate the target distribution $\pi$, but samples do not come directly from $\pi$. For nonnegative functions $p_1, p_2$ defined on $\mathcal{X}$, let $p_1 p_2$ denote the distribution obtained by multiplying weights $p_1(x)$ and $p_2(x)$ at every point and renormalizing:

$$p_1 p_2(x) = \frac{p_1(x)p_2(x)}{\sum_{x'} p_1(x')p_2(x')}.$$

Samples $x_1, \ldots, x_m$ come from the *biased distribution* $\pi s$ where $s$ is the *sampling distribution*. This setup corresponds to the situation when an event being observed occurs at the point $x$ with probability $\pi(x)$ while we perform an independent observation with probability $s(x)$. The probability of observing an event at $x$ given that we observe an event is then equal to $\pi s(x)$. The empirical distribution of $m$ samples drawn from $\pi s$ will be denoted by $\widetilde{\pi s}$. We assume that $s$ is known (principal assumption, see introduction) and strictly positive (technical assumption).

**Approach I: Debiasing Averages.** In our first approach, we use the same algorithm as for the unbiased case but employ a different method to obtain confidence intervals $[a_j, b_j]$. Since we do not have direct access to samples from $\pi$, we use a version of the Bias Correction Theorem of Zadrozny [6] to convert expectations with respect to $\pi s$ to expectations with respect to $\pi$.

**Theorem 2** (Bias Correction Theorem [6], Translation Theorem [7]).

$$\pi s[\boldsymbol{f}/s]\big/\pi s[1/s] = \pi[\boldsymbol{f}].$$

Hence, it suffices to give confidence intervals for $\pi s[\boldsymbol{f}/s]$ and $\pi s[1/s]$ to obtain confidence intervals for $\pi[\boldsymbol{f}]$.

**Corollary 3.** *Assume that for some sample-derived bounds $c_j, d_j, 0 \leq j \leq n$, with high probability $0 < c_0 \leq \pi s[1/s] \leq d_0$ and $0 \leq c_j \leq \pi s[f_j/s] \leq d_j$ for all $1 \leq j \leq n$. Then with at least the same probability $c_j/d_0 \leq \pi[f_j] \leq d_j/c_0$ for all $1 \leq j \leq n$.*

If $s$ is bounded away from 0 then Chernoff bounds may be used to determine $c_j, d_j$. Corollary 3 and Theorem 1 then yield guarantees that this method's performance converges, with increasing sample sizes, to that of the "best" Gibbs distribution.

In practice, confidence intervals $[c_j, d_j]$ may be determined using expressions analogous to (5) and (7) for random variables $f_j/s$, $1/s$ and the empirical distribution $\widetilde{\pi s}$. After first restricting the confidence intervals in a natural fashion, this yields Algorithm 1. Alternatively, we could use bootstrap or other types of estimates for the confidence intervals.

**Approach II: Factoring Bias Out.** The second algorithm does not approximate $\pi$ directly, but uses maxent to estimate the distribution $\pi s$ and then converts this estimate into an approximation of $\pi$. If the default estimate of $\pi$ is $q_0$, then the default estimate of $\pi s$ is $q_0 s$. Applying unbiased maxent to the empirical distribution $\widetilde{\pi s}$ with the default $q_0 s$, we obtain a $q_0 s$-Gibbs distribution $q_0 s e^{\hat{\boldsymbol{\lambda}} \cdot \boldsymbol{f}}$ approximating $\pi s$. We factor out $s$ to obtain $q_0 e^{\hat{\boldsymbol{\lambda}} \cdot \boldsymbol{f}}$ as an estimate of $\pi$. This yields the algorithm FACTORBIASOUT.

This approach corresponds to $\ell_1$-regularized maximum likelihood estimation of $\pi$ by $q_0$-Gibbs distributions. When $\pi$ itself is $q_0$-Gibbs then the distribution $\pi s$ is $q_0 s$-Gibbs. Performance guarantees for unbiased maxent imply that estimates of $\pi s$ converge to $\pi s$ as the number of samples increases. Now, if $\inf_x s(x) > 0$ (which is the case for finite $\mathcal{X}$) then estimates of $\pi$ obtained by factoring out $s$ converge to $\pi$ as well.

When $\pi$ is not $q_0$-Gibbs then $\pi s$ is not $q_0 s$-Gibbs either. We approximate $\pi$ by a $q_0$-Gibbs distribution $\hat{q} = q_{\hat{\boldsymbol{\lambda}}}$ which, with an increasing number of samples, minimizes $\mathrm{RE}(\pi s \parallel q_{\boldsymbol{\lambda}} s)$ rather than $\mathrm{RE}(\pi \parallel q_{\boldsymbol{\lambda}})$. Our next example shows that these two minimizers may be different.

**Example 1.** Consider the space $\mathcal{X} = \{1, 2, 3, 4\}$ with two features $f_1, f_2$. Features $f_1, f_2$, target distribution $\pi$, sampling distribution $s$ and the biased distribution $\pi s$ are given in Table 1. We use the uniform distribution as a default estimate. The minimizer of $\mathrm{RE}(\pi \parallel q_{\boldsymbol{\lambda}})$ is the unique uniform-Gibbs distribution $q^*$ such that $q^*[\boldsymbol{f}] = \pi[\boldsymbol{f}]$. Similarly, the minimizer $q^{**}s$ of $\mathrm{RE}(\pi s \parallel q_{\boldsymbol{\lambda}} s)$ is the unique $s$-Gibbs distribution for which $q^{**}s[\boldsymbol{f}] = \pi s[\boldsymbol{f}]$. Solving for these exactly, we find that $q^*$ and $q^{**}$ are as given in Table 1, and that these two distributions differ. $\square$

Even though FACTORBIASOUT does not minimize $\mathrm{RE}(\pi \parallel q_{\boldsymbol{\lambda}})$, we can show that it minimizes a different Bregman divergence. More precisely, it minimizes a Bregman divergence between certain projections of the two distributions. Bregman divergences generalize some common distance measures such as relative entropy or the squared Euclidean distance, and enjoy many of the same favorable properties. The Bregman divergence associated with a convex function $F$ is defined as $\mathrm{D}_F(u \parallel v) = F(u) - F(v) - \nabla F(v) \cdot (u - v)$.

**Proposition 4.** *Define $F : \mathbb{R}_+^{\mathcal{X}} \to \mathbb{R}$ as $F(u) = \sum_x s(x)u(x)\log u(x)$. Then $F$ is a convex function and for all $p_1, p_2 \in \Delta$, $\mathrm{RE}(p_1 s \parallel p_2 s) = \mathrm{D}_F(p'_1 \parallel p'_2)$, where $p'_1(x) = p_1(x)/\sum_{x'} s(x')p_1(x')$ and $p'_2(x) = p_2(x)/\sum_{x'} s(x')p_2(x')$ are projections of $p_1, p_2$ along lines $tp, t \in \mathbb{R}$ onto the hyperplane $\sum_x s(x)p(x) = 1$.*

**Approach III: Approximating FACTORBIASOUT.** As mentioned in the introduction, knowing the sampling distribution $s$ exactly is unrealistic. However, we often have access to samples from $s$. In this approach we assume that $s$ is unknown but that, in addition to samples $x_1, \ldots, x_m$ from $\pi s$, we are also given a separate set of samples $x_{(1)}, x_{(2)}, \ldots, x_{(N)}$ from $s$. We use the algorithm FACTORBIASOUT with the sampling distribution $s$ replaced by the corresponding empirical distribution $\tilde{s}$.

To simplify the algorithm, we note that instead of using $q_0\tilde{s}$ as a default estimate for $\pi s$, it suffices to replace the sample space $\mathcal{X}$ by $\mathcal{X}' = \{x_{(1)}, x_{(2)}, \ldots, x_{(N)}\}$ and use $q_0$

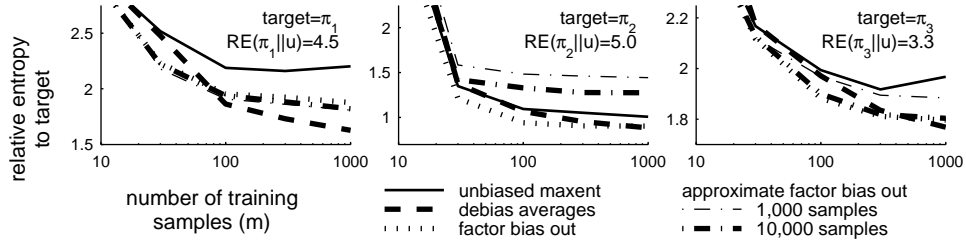

Figure 1: *Learning curves for synthetic experiments.* We use $u$ to denote the uniform distribution. For the sampling distribution $s$, $\text{RE}(s \parallel u) = 0.8$. Performance is measured in terms of relative entropy to the target distribution as a function of an increasing number of training samples. The number of samples is plotted on a log scale.

restricted to $\mathcal{X}'$ as a default. The last step of factoring out $\tilde{s}$ is equivalent to using $\hat{\boldsymbol{\lambda}}$ returned for space $\mathcal{X}'$ on the entire space $\mathcal{X}$.

When the sampling distribution $s$ is correlated with feature values, $\mathcal{X}'$ might not cover all feature ranges. In that case, reprojecting on $\mathcal{X}$ may yield poor estimates outside of these ranges. We therefore do "clamping", restricting values $f_j(x)$ to their ranges over $\mathcal{X}'$ and capping values of the exponent $\hat{\boldsymbol{\lambda}} \cdot \boldsymbol{f}(x)$ at its maximum over $\mathcal{X}'$. The resulting algorithm is called APPROXFACTORBIASOUT.

## 4 Experiments

Conducting real data experiments to evaluate bias correction techniques is difficult, because bias is typically unknown and samples from unbiased distributions are not available. Therefore, synthetic experiments are often a necessity for precise evaluation. Nevertheless, in addition to synthetic experiments, we were also able to conduct experiments with real-world data for habitat modeling.

**Synthetic experiments.** In synthetic experiments, we generated three target uniform-Gibbs distributions $\pi_1$, $\pi_2$, $\pi_3$ over a domain $\mathcal{X}$ of size 10,000. These distributions were derived from 65 features indexed as $f_i, 0 \leq i \leq 9$ and $f_{ij}, 0 \leq i \leq j \leq 9$. Values $f_i(x)$ were chosen independently and uniformly in $[0, 1]$, and we set $f_{ij}(x) = f_i(x)f_j(x)$. Fixing these features, we generated weights for each distribution. Weights $\lambda_i$ and $\lambda_{ii}$ were generated jointly to capture a range of different behaviors for values of $f_i$ in the range $[0, 1]$.

Let $U_{\mathcal{S}}$ denote a random variable uniform over the set $\mathcal{S}$. Each instance of $U_{\mathcal{S}}$ corresponds to a new independent variable. We set $\lambda_{ii} = U_{\{-1,0,1\}}U_{[1,5]}$ and $\lambda_i$ to be $\lambda_{ii}U_{[-3,1]}$ if $\lambda_{ii} \neq 0$, and $U_{\{-1,1\}}U_{[2,10]}$ otherwise. Weights $\lambda_{ij}, i < j$ were chosen to create correlations between $f_i$'s that would be observable, but not strong enough to dominate $\lambda_i$'s and $\lambda_{ii}$'s. We set $\lambda_{ij} = -0.5$ or $0$ or $0.5$ with respective probabilities 0.05, 0.9 and 0.05. In maxent, we used a subset of features specifying target distributions and some irrelevant features. We used features $f'_i, 0 \leq i \leq 9$ and their squares $f'_{ii}$, where $f'_i(x) = f_i(x)$ for $0 \leq i \leq 5$ (relevant features) and $f'_i(x) = U_{[0,1]}$ for $6 \leq i \leq 9$ (irrelevant features). Once generated, we used the same set of features in all experiments. We generated a sampling distribution $s$ correlated with target distributions. More specifically, $s$ was a Gibbs distribution generated from features $f_i^{(s)}, 0 \leq i \leq 5$ and their squares $f_{ii}^{(s)}$, where $f_i^{(s)}(x) = U_{[0,1]}$ for $0 \leq i \leq 1$ and $f_i^{(s)} = f_{i+2}$ for $2 \leq i \leq 5$. We used weights $\lambda_i^{(s)} = 0$ and $\lambda_{ii}^{(s)} = -1$.

For every target distribution, we evaluated the performance of UNBIASEDMAXENT, DEBIASAVERAGES, FACTORBIASOUT and APPROXFACTORBIASOUT with 1,000 and 10,000 samples from the sampling distribution. The performance was evaluated in terms of relative entropy to the target distribution. We used training sets of sizes 10 to 1000. We considered five randomly generated training sets and took the average performance over these five sets for settings of $\beta$ from the range $[0.05, 4.64]$. We report results for the best $\beta$, chosen separately for each average. The rationale behind this approach is that we want to

Table 2: *Results of real data experiments.* Average performance of unbiased maxent and three bias correction approaches over all species in six regions. The uniform distribution would receive the log loss of 14.2 and AUC of 0.5. Results of bias correction approaches are italicized if they are significantly worse and set in boldface if they are significantly better than those of the unbiased maxent according to a paired $t$-test at the level of significance 5%.

| | average log loss | | | | | | average AUC | | | | | |
|---|---|---|---|---|---|---|---|---|---|---|---|---|
| | *awt* | *can* | *nsw* | *nz* | *sa* | *swi* | *awt* | *can* | *nsw* | *nz* | *sa* | *swi* |
| unbiased maxent | 13.78 | 12.89 | 13.40 | 13.77 | 13.14 | 12.81 | 0.69 | 0.58 | 0.71 | 0.72 | 0.78 | 0.81 |
| debias averages | *13.92* | 13.10 | 13.88 | *14.31* | *14.10* | *13.59* | 0.67 | 0.64 | *0.65* | *0.67* | *0.68* | *0.78* |
| factor bias out | 13.90 | *13.13* | *14.06* | *14.20* | *13.66* | *13.46* | 0.71 | **0.69** | 0.72 | 0.72 | 0.78 | **0.83** |
| apx. factor bias out | 13.89 | *13.40* | *14.19* | *14.07* | *13.62* | *13.41* | **0.72** | **0.72** | 0.73 | 0.73 | 0.78 | **0.84** |

explore the potential performance of each method.

Figure 1 shows the results at the optimal $\beta$ as a function of an increasing number of samples. FACTORBIASOUT is always better than UNBIASEDMAXENT. DEBIASAVERAGES is worse than UNBIASEDMAXENT for small sample sizes, but as the number of training samples increases, it soon outperforms UNBIASEDMAXENT and eventually also outperforms FACTORBIASOUT. APPROXFACTORBIASOUT improves as the number of samples from the sampling distribution increases from 1,000 to 10,000, but both versions of APPROX-FACTORBIASOUT perform worse than UNBIASEDMAXENT for the distribution $\pi_2$.

**Real data experiments.** In this set of experiments, we evaluated maxent in the task of estimating species habitats. The sample space is a geographic region divided into a grid of cells and samples are known occurrence localities — cells where a given species was observed. Every cell is described by a set of *environmental variables*, which may be categorical, such as vegetation type, or continuous, such as altitude or annual precipitation. Features are real-valued functions derived from environmental variables. We used *binary indicator features* for different values of categorical variables and *binary threshold features* for continuous variables. The latter are equal to one when the value of a variable is greater than a fixed threshold and zero otherwise.

Species sample locations and environmental variables were all produced and used as part of the "Testing alternative methodologies for modeling species' ecological niches and predicting geographic distributions" Working Group at the National Center for Ecological Analysis and Synthesis (NCEAS). The working group compared modeling methods across a variety of species and regions. The training set contained presence-only data from unplanned surveys or incidental records, including those from museums and herbariums. The test set contained presence-absence data from rigorously planned independent surveys.

We compared performance of our bias correction approaches with that of the unbiased maxent which was among the top methods in the NCEAS comparison [13]. We used the full dataset consisting of 226 species in 6 regions with 2–5822 training presences per species (233 on average) and 102–19120 test presences/absences. For more details see [13].

We treated training occurrence locations for all species in each region as sampling distribution samples and used them directly in APPROXFACTORBIASOUT. In order to apply DEBIASAVERAGES and FACTORBIASOUT, we estimated the sampling distribution using unbiased maxent. Sampling distribution estimation is also the first step of [6]. In contrast with that work, however, our experiments do not use the sampling distribution estimate during evaluation and hence do not depend on its quality.

The resulting distributions were evaluated on test presences according to the log loss and on test presences and absences according to the *area under an ROC curve* (AUC) [14]. AUC quantifies how well the predicted distribution ranks test presences above test absences. Its value is equal to the probability that a randomly chosen presence will be ranked above a randomly chosen absence. The uniformly random prediction receives AUC of 0.5 while a perfect prediction receives AUC of 1.0.

In Table 2 we show performance of our three approaches compared with the unbiased maxent. All three algorithms yield on average a worse log loss than the unbiased maxent. This can perhaps be attributed to the imperfect estimate of the sampling distribution or to

the sampling distribution being zero over large portions of the sample space. In contrast, when the performance is measured in terms of AUC, FACTORBIASOUT and APPROX-FACTORBIASOUT yield on average the same or better AUC as UNBIASEDMAXENT in all six regions. Improvements in regions *awt*, *can* and *swi* are dramatic enough so that both of these methods perform better than any method evaluated in [13].

## 5   Conclusions

We have proposed three approaches that incorporate information about sample selection bias in maxent and demonstrated their utility in synthetic and real data experiments. Experiments also raise several questions that merit further research: DEBIASAVERAGES has the strongest performance guarantees, but it performs the worst in real data experiments and catches up with other methods only for large sample sizes in synthetic experiments. This may be due to poor estimates of unbiased confidence intervals and could be possibly improved using a different estimation method. FACTORBIASOUT and APPROXFACTOR-BIASOUT improve over UNBIASEDMAXENT in terms of AUC over real data, but are worse in terms of log loss. This disagreement suggests that methods which aim to optimize AUC directly could be more successful in species modeling, possibly incorporating some concepts from FACTORBIASOUT and APPROXFACTORBIASOUT. APPROXFACTORBIAS-OUT performs the best on real world data, possibly due to the direct use of samples from the sampling distribution rather than a sampling distribution estimate. However, this method comes without performance guarantees and does not exploit the knowledge of the full sample space. Proving performance guarantees for APPROXFACTORBIASOUT remains open for future research.

### Acknowledgments

This material is based upon work supported by NSF under grant 0325463. Any opinions, findings, and conclusions or recommendations expressed in this material are those of the authors and do not necessarily reflect the views of NSF. The NCEAS data was kindly shared with us by the members of the "Testing alternative methodologies for modeling species' ecological niches and predicting geographic distributions" Working Group, which was supported by the National Center for Ecological Analysis and Synthesis, a Center funded by NSF (grant DEB-0072909), the University of California and the Santa Barbara campus.

### References

[1] Jane Elith. Quantitative methods for modeling species habitat: Comparative performance and an application to Australian plants. In Scott Ferson and Mark Burgman, editors, *Quantitative Methods for Conservation Biology*, pages 39–58. Springer-Verlag, 2002.

[2] A. Guisan and N. E. Zimmerman. Predictive habitat distribution models in ecology. *Ecological Modelling*, 135:147–186, 2000.

[3] Steven J. Phillips, Miroslav Dudík, and Robert E. Schapire. A maximum entropy approach to species distribution modeling. In *Proceedings of the Twenty-First International Conference on Machine Learning*, 2004.

[4] S. Reddy and L. M. Dávalos. Geographical sampling bias and its implications for conservation priorities in Africa. *Journal of Biogeography*, 30:1719–1727, 2003.

[5] Barbara R. Stein and John Wieczorek. Mammals of the world: MaNIS as an example of data integration in a distributed network environment. *Biodiversity Informatics*, 1(1):14–22, 2004.

[6] Bianca Zadrozny. Learning and evaluating classifiers under sample selection bias. In *Proceedings of the Twenty-First International Conference on Machine Learning*, 2004.

[7] Bianca Zadrozny, John Langford, and Naoki Abe. Cost-sensitive learning by cost-proportionate example weighting. In *Proceedings of the Third IEEE International Conference on Data Mining*, 2003.

[8] James J. Heckman. Sample selection bias as a specification error. *Econometrica*, 47(1):153–161, 1979.

[9] Robert M. Groves. *Survey Errors and Survey Costs*. Wiley, 1989.

[10] Roderick J. Little and Donald B. Rubin. *Statistical Analysis with Missing Data*. Wiley, second edition, 2002.

[11] Charles Elkan. The foundations of cost-sensitive learning. In *Proceedings of the Seventeenth International Joint Conference on Artificial Intelligence*, 2001.

[12] Miroslav Dudík, Steven J. Phillips, and Robert E. Schapire. Performance guarantees for regularized maximum entropy density estimation. In *17th Annual Conference on Learning Theory*, 2004.

[13] J. Elith, C. Graham, and NCEAS working group. Comparing methodologies for modeling species' distributions from presence-only data. In preparation.

[14] J. A. Hanley and B. S. McNeil. The meaning and use of the area under a receiver operating characteristic (ROC) curve. *Radiology*, 143:29–36, 1982.
